# Image Parsing via Stochastic Scene Grammar

**Yibiao Zhao**[*]
Department of Statistics
University of California, Los Angeles
Los Angeles, CA 90095
ybzhao@ucla.edu

**Song-Chun Zhu**
Department of Statistics and Computer Science
University of California, Los Angeles
Los Angeles, CA 90095
sczhu@stat.ucla.edu

## Abstract

This paper proposes a parsing algorithm for scene understanding which includes four aspects: computing 3D scene layout, detecting 3D objects (e.g. furniture), detecting 2D faces (windows, doors etc.), and segmenting background. In contrast to previous scene labeling work that applied discriminative classifiers to pixels (or super-pixels), we use a generative **Stochastic Scene Grammar** (SSG). This grammar represents the compositional structures of visual entities from scene categories, 3D foreground/background, 2D faces, to 1D lines. The grammar includes three types of production rules and two types of contextual relations. **Production rules**: (i) AND rules represent the decomposition of an entity into sub-parts; (ii) OR rules represent the switching among sub-types of an entity; (iii) SET rules represent an ensemble of visual entities. **Contextual relations**: (i) *Cooperative* "+" relations represent positive links between binding entities, such as hinged faces of a object or aligned boxes; (ii) *Competitive* "-" relations represents negative links between competing entities, such as mutually exclusive boxes. We design an efficient MCMC inference algorithm, namely **Hierarchical cluster sampling**, to search in the large solution space of scene configurations. The algorithm has two stages: (i) *Clustering*: It forms all possible higher-level structures (clusters) from lower-level entities by production rules and contextual relations. (ii) *Sampling*: It jumps between alternative structures (clusters) in each layer of the hierarchy to find the most probable configuration (represented by a parse tree). In our experiment, we demonstrate the superiority of our algorithm over existing methods on public dataset. In addition, our approach achieves richer structures in the parse tree.

## 1 Introduction

Scene understanding is an important task in neural information processing systems. By analogy to natural language parsing, we pose the scene understanding problem as parsing an image into a hierarchical structure of visual entities (in Fig.1(i)) using the Stochastic Scene Grammar (SSG). The literature of scene parsing can be categorized into two categories: discriminative approaches and generative approaches.

**Discriminative approaches** focus on classifying each pixel (or superpixel) to a semantic label (building, sheep, road, boat etc.) by discriminative Conditional Random Fields (CRFs) model [5]-[7]. Without an understanding of the scene structure, the pixel-level labeling is insufficient to represent the knowledge of object occlusions, 3D relationships, functional space etc. To address this problem, geometric descriptions were added to the scene interpretation. Hoiem et al. [1] and Saxena et al. [8] generated the surface orientation labels and the depth labels by exploring rich geometric

---

[*]http://www.stat.ucla.edu/~ybzhao/research/sceneparsing

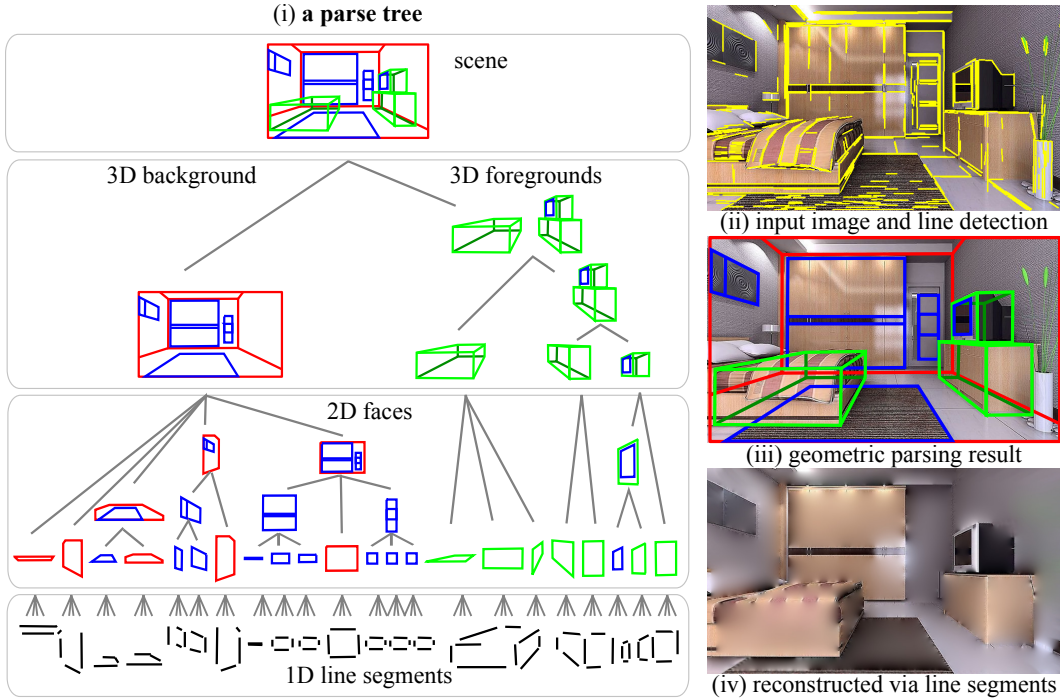

Figure 1: A parse tree of geometric parsing result.

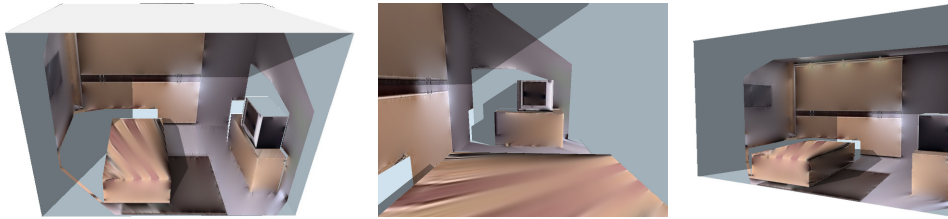

Figure 2: 3D synthesis of novel views based on the parse tree.

features and context information. Gupta et al. [9] posed the 3D objects as blocks and infers its 3D properties such as occlusion, exclusion and stableness in addition to surface orientation labels. They showed the global 3D prior does help the 2D surface labeling. For the indoor scene, Hedau et al. [2], Wang et al. [3] and Lee et al. [4] adopted different approaches to model the geometric layout of the background and/or foreground objects, and fit their models into Structured SVM (or Latent SVM) settings [10]. The Structured SVM uses features extracted jointly from input-output pairs and maximizes the margin over the structured output space. These algorithms involve hidden variables or structured labels in discriminative training. However, these discriminative approaches lack a general representation of visual vocabulary and a principled approach for exploring the compositional structure.

**Generative approaches** make efforts to model the reconfigurable graph structures in generative probabilistic models. The stochastic grammar were used to parse natural languages [11]. Compositional models for the hierarchical structure and sharing parts were studied in visual object recognition [12]-[15]. Zhu and Mumford [16] proposed an AND/OR Graph Model to represent the compositional structures in vision. However, the expressive power of configurable graph structures comes at the cost of high computational complexity of searching in a large configuration space. In order to accelerate the inference, the Adaptor Grammars [17] applied an idea of "adaptor" (re-using subtree) that induce dependencies among successive uses. Han and Zhu [18] applied grammar rules, in a greedy manner, to detect rectangular structures in man-made scenes. Porway et al. [19] [20]allowed the Markov chain jumping between competing solutions by a C4 algorithm.

**Overview of the approach**. In this paper, we parse an image into a hierarchical structure, namely *a parse tree* as shown in Fig.1. The parse tree covers a wide spectrum of visual entities, including scene categories, 3D foreground/background, 2D faces, and 1D line segments. With the low-level information of the parse tree, we reconstruct the original image by the appearance of line segments, as shown in Fig.1(iv). With the high-level information of the parse tree, we further recover the 3D scene by the geometry of 3D background and foreground objects, as shown in Fig.2.

This paper has two major contributions to the scene parsing problems:

**(I)** A Stochastic Scene Grammar (SSG) is introduced to represent the hierarchical structure of visual entities. The grammar starts with a single root node (the scene) and ends with a set of terminal nodes (line segments). In between, we generate all intermediate 3D/2D sub-structures by three types of production rules and two types of contextual relations, as illustrated in Fig.3. **Production rules**: *AND*, *OR*, and *SET*. (i) The AND rule encodes how sub-parts are composed into a larger structure. For example, three hinged faces form a 3D box, four linked line segments form a rectangle, a background and inside objects form a scene in Fig.3(i); (ii) The SET rule represents an ensemble of entities, e.g. a set of 3D boxes or a set of 2D regions as in Fig.3(ii); (iii)The OR rule represents a switch between different sub-types, e.g. a 3D foreground and 3D background have several switchable types in Fig.3(iii). **Contextual relations**: *Cooperative* "+" and *Competitive* "-". (i) If the visual entities satisfy a cooperative "+" relation, they tend to bind together, e.g. hinged faces of a foreground box showed in Fig.3(a). (ii) If entities satisfy a competitive "-" relation, they compete with each other for presence, e.g. two exclusive foreground boxes competing for a same space in Fig.3(b).

**(II)** A *hierarchical cluster sampling* algorithm is proposed to perform inference efficiently in SSG model. The algorithm accelerates a Markov chain search by exploring contextual relations. It has two stages: (i) **Clustering**. Based on the detected line segments in Fig.1(ii), we form all possible larger structures (clusters). In each layer, the entities are first filtered by the Cooperative "+" constraints, they then form a cluster only if they satisfy the "+" constraints, e.g. several faces form a cluster of a box when their edges are hinged tightly. (ii) **Sampling**. The sampling process makes a big reversible jumps by switching among competing sub-structures (e.g. two exclusive boxes).

In summary, the Stochastic Scene Grammar is a general framework to parse a scene with a large number of geometric configurations. We demonstrate the superiority of our algorithm over existing methods in the experiment.

## 2   Stochastic Scene Grammar

The Stochastic Scene Grammar (SSG) is defined as a four-tuple $G = (S, V, R, P)$, where $S$ is a start symbol at the root (scene); $V = V^N \cup V^T$, $V^N$ is a finite set of non-terminal nodes (structures or sub-structures), $V^T$ is a finite set of terminal nodes (line segments); $R = \{r : \alpha \rightarrow \beta\}$ is a set of production rules, each of which represents a generating process from a parent node $\alpha$ to its child nodes $\beta = Ch_\alpha$. $P(r) = P(\beta|\alpha)$ is an expansion probability for each production rule $(r : \alpha \rightarrow \beta)$. A set of all valid configurations $C$ derived from production rules is called a *language*:

$$L(G) = \{C : S \xrightarrow{\{r_i\}} C, \{r_i\} \subset R, C \subset V^T, P(\{r_i\}) > 0\}.$$

**Production rules**. We define three types of stochastic production rules $R^{AND}$, $R^{OR}$, $R^{SET}$ to represent the structural *regularity* and *flexibility* of visual entities. The regularity is enforced by the AND rule and the flexibility is expressed by the OR rule. The SET rule is a mixture of OR and AND rules.

(i) An AND rule $(r^{AND} : A \rightarrow a \cdot b \cdot c)$ represents the *decomposition* of a parent node $A$ into three sub-parts $a$, $b$, and $c$. The probability $P(a, b, c|A)$ measures the compatibility (contextual relations) among sub-structures $a, b, c$. As seen Fig.3(i), the grammar outputs a high probability if the three faces of a 3D box are well hinged, and a low probability if the foreground box lays out of the background.

(ii) An OR rule $(r^{OR} : A \rightarrow a \mid b)$ represents the *switching* between two sub-types $a$ and $b$ of a parent node $A$. The probability $P(a|A)$ indicates the preference for one subtype over others. For 3D foreground in Fig.3(iii), the three sub-types in the third row represent objects below the horizon. These objects appear with high probabilities. Similarly, for the 3D background in Fig.3(iii), the camera rarely faces the ceiling or the ground, hence, the three sub-types in the middle row have

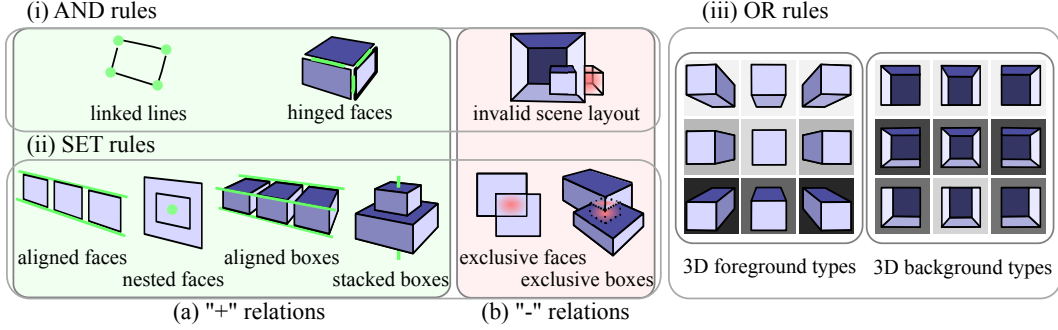

(a) "+" relations      (b) "-" relations

Figure 3: Three types of production rules: AND (i), SET (ii) OR (iii), and two types of contextual relations: cooperative "+" relations (a), competitive "-" relations (b).

higher probabilities (the higher the darker). Moreover, OR rules also model the discrete size of entities, which is useful to rule out the extreme large or small entities.

(iii) An SET rule ($r^{SET} : A \rightarrow \{a\}_k, k \geq 0$) represents an *ensemble* of $k$ visual entities. The SET rule is equivalent to a mixture of OR and AND rules ($r^{SET} : A \rightarrow \emptyset \mid a \mid a \cdot a \mid a \cdot a \cdot a \mid \cdots$). It first chooses a set size $k$ by ORing, and forms an ensemble of $k$ entities by ANDing. It is worth noting that the OR rule essentially changes the graph topology of the output parse tree by changing its node size $k$. In this way, as seen in Fig.3(ii), the SET rule generates a set of 3D/2D entities which satisfy some contextual relations.

**Contextual relations**. There are two kinds of contextual relations, *Cooperative* "+" relations and *Competitive* "-" relations, which involve in the AND and SET rules.

(i) The cooperative "+" relations specify the *concurrent* patterns in a scene, e.g. hinged faces, nested rectangle, aligned windows in Fig.3(a). The visual entities satisfying a cooperative "+" relation tend to bind together.

(i) The competitive "-" relations specify the *exclusive* patterns in a scene. If entities satisfy competitive "-" relations, they compete with each other for the presence. As shown in Fig.3(b), if a 3D box is not contained by its background, or two 2D/3D objects are exclusive with one another, these cases will rarely be in a solution simultaneously.

The *tight structures* vs. the *loose structure*: If several visual entities satisfy a cooperative "+" relation, they tend to bind together, and we call them *tight structures*. These tight structures are grouped into clusters in the early stage of inference (Sect.4). If the entities neither satisfy any cooperative "+" relations nor violate a competitive "-" relation, they may be loosely combined. We call them *loose structures*, whose combinations are sampled in a later stage of inference (Sect.4). With the three production rules and two contextual relations, SSG is able to handle an enormous number of configurations and large geometric variations, which are the major difficulties in our task.

## 3 Bayesian formulation of the grammar

We define a posterior distribution for a solution (a parse tree) $pt$ conditioned on an input image $I$. This distribution is specified in terms of the statistics defined over the derivation of production rules.

$$P(pt|I) \propto P(pt)P(I|pt) = P(S) \prod_{v \in V^N} P(Ch_v|v) \prod_{v \in V^T} P(I|v) \tag{1}$$

where $I$ is the input image, $pt$ is the parse tree. The probability derivation represents a generating process of the production rules $\{r : v \rightarrow Ch_v\}$ from the start symbol $S$ to the nonterminal nodes $v \in V^N$, and to the children of non-terminal nodes $Ch_v$. The generating process stops at the terminal nodes $v \in V^T$ and generates the image $I$.

We use a probabilistic graphical model of AND/OR graph [12, 17] to formulate our grammar. The graph structure $G = (V, E)$ consists of a set of nodes $V$ and a set of edges $E$. The edge define a

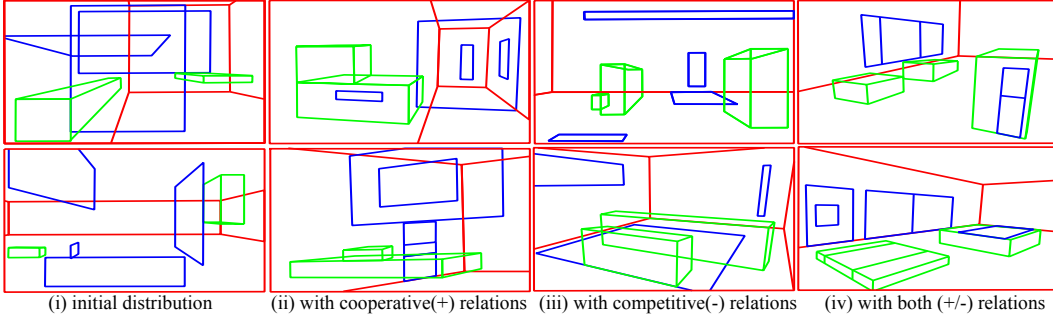

| (i) initial distribution | (ii) with cooperative(+) relations | (iii) with competitive(-) relations | (iv) with both (+/-) relations |

Figure 4: Learning to synthesize. (a)-(d) Some typical samples drawn from Stochastic Scene Grammar model with/without contextual relations.

parent-child conditional dependency for each production rule. The posterior distribution of a parse graph $pt$ is given by a family of Gibbs distributions: $P(pt|I; \lambda) = 1/Z(I; \lambda) \exp\{-E(pt|I)\}$, where $Z(I; \lambda) = \sum_{pt \in \Omega} \exp\{-E(pt|I)\}$ is a partition function summation over the solution space $\Omega$.

The energy is decomposed into three potential terms:

$$E(pt|I) = \sum_{v \in V^{OR}} E^{OR}(A_T(Ch_v)) + \sum_{v \in V^{AND}} E^{AND}(A_G(Ch_v)) + \sum_{\Lambda_v \in \Lambda_I, v \in V^T} E^T(I(\Lambda_v)) \quad (2)$$

(i) **The energy for OR nodes** is defined over "type" attributes $A_T(Ch_v)$ of ORing child nodes. The potential captures the prior statistics on each switching branch. $E^{OR}(A_T(v)) = -\log P(v \to A_T(v)) = -\log\{\frac{\#(v \to A_T(v))}{\sum_{u \in Ch(v)} \#(v \to u)}\}$. The switching probability of foreground objects and the background layout is shown in Fig.3(iii).

(ii) **The energy for AND nodes** is defined over "geometry" attribute $A_G(Ch_v)$ of ANDing child nodes. They are Markov Random Fields (MRFs) inside a tree-structure. We define both "+" relations and "-" relations as $E^{AND} = \lambda^+ h^+(A_G(Ch_v)) + \lambda^- h^-(A_G(Ch_v))$, where $h(*)$ are sufficient statistics in the exponential model, $\lambda$ are their parameters. For 2D faces as an example, the "+" relation specifies a quadratic distance between their connected joints $h^+(A_G(Ch_v)) = \sum_{a,b \in Ch_v}(X(a) - X(b))^2$, and the "-" relation specifies an overlap rate between their occupied image area $h^-(A_G(Ch_v)) = (\Lambda_a \cap \Lambda_b)/(\Lambda_a \cup \Lambda_b)$, $a, b \in Ch_v$.

(iii) **The energy for Terminal nodes** is defined over bottom-up image features $I(\Lambda_v)$ on the image area $\Lambda_v$. The features used in this paper include: (a) surface labels of geometric context [1], (b) a 3D orientation map [21], (c) the MDL coding length of line segments [20]. This term only captures the features from their dominant image area $\Lambda_v$, and avoids the double counting of the shared edges and the occluded areas.

We learn the context-sensitive grammar model of SSG from a context-free grammar. Under the learning framework of minimax entropy [25], we enforce the contextual relations by adding statistical constraints sequentially. The learning process matches the statistics between the current distribution $p$ and a targeted distribution $f$ by adding the most violated constraint in each iteration. Fig.4 shows the typical samples drawn from the learned SSG model. With more contextual relations being added, the sampled configurations become more similar to a real scene, and the statistics of the learned distribution become closer to that of target distribution.

## 4 Inference with hierarchical cluster sampling

We design a hierarchical cluster sampling algorithm to infer the optimal parse tree for the SSG model. A parse tree specifies a configuration of visual entities. The combination of configurations makes the solution space expand exponentially, and it is NP-hard to enumerate all parse trees in such a large space.

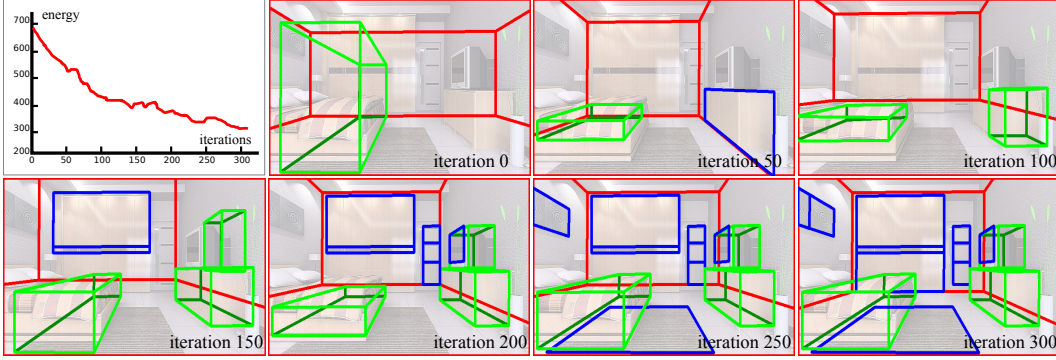

Figure 5: The hierarchical cluster sampling process.

In order to detecting scene components, neither sliding window (top-down) nor binding (bottom-up) approaches can handle the large geometric variations and an enormous number of configurations. In this paper we combine the bottom-up and top-down process by exploring the contextual relations defined on the grammar model. The algorithm first perform a bottom-up clustering stage and follow by a top-down sampling stage.

**In the clustering stage**, we group visual entities into clusters (tight structures) by filtering the entities based on cooperative "+" relations. With the low-level line segments as illustrated in Fig.1.(iv), we detect substructures, such as 2D faces, aligned and nested 2D faces, 3D boxes, aligned and stacked 3D boxes (in Fig.3(a)) layer by layer. The clusters $Cl$ are formed only if the cooperative "+" constraints are satisfied. The proposal probability for each cluster $Cl$ is defined as

$$P_+(Cl|I) = \prod_{v \in Cl^{OR}} P^{OR}(A_T(v)) \prod_{u,v \in Cl^{AND}} P_+^{AND}(A_G(u), A_G(v)) \prod_{v \in Cl^T} P^T(I(\Lambda_v)). \qquad (3)$$

Clusters with marginal probabilities below threshold are pruned. The threshold is learned by a probably approximately admissible (PAA) bound [23]. The clusters so defined are enumerable.

**In the sampling stage**, we performs an efficient MCMC inference to search in the combinational space. In each step, the Markov chain jumps over a cluster (a big set of nodes) given information of "what goes together" from clustering. The algorithm proposes a new parse tree: $pt^* = pt + Cl^*$ with the cluster $Cl^*$ conditioning on the current parse tree $pt$. To avoid heavy computation, the proposal probability is defined as

$$Q(pt^*|pt, I) = P_+(Cl^*|I) \prod_{u \in Cl^{AND}, v \in pt^{AND}} P_-^{AND}(A_G(u)|A_G(v)). \qquad (4)$$

The algorithm gives more weights to the proposals with strong bottom-up support and tight "+" relations by $P_+(Cl|I)$, and simultaneously avoids the exclusive proposals with "-" relations by $P_-^{AND}(A_G(u)|A_G(v))$. All of these probabilities are pre-computed before sampling. The marginal probability of each cluster $P_+(Cl|I)$ is computed during the clustering stage, and the probability for each pair-wise negative "-" relations $P_-^{AND}(A_G(u)|A_G(v))$ is then calculated and stored in a look-up table. The algorithm also proposes a new parse tree by pruning current parse tree randomly.

By applying the Metropolis-Hastings acceptance probability $\alpha(pt \to pt*) = min\{1, \frac{Q(pt|pt*,I)}{Q(pt*|pt,I)} \cdot \frac{P(pt*|I)}{P(pt|I)}\}$, the Markov chain search satisfies the detailed balance principle, which implies that the Markov chain search will converge to the global optimum in Fig.5.

## 5   Experiments

We evaluate our algorithm on both the UIUC indoor dataset [2] and our own dataset. The UIUC dataset contains 314 cluttered indoor images, of which the ground-truth is two label maps of background layout with/without foreground objects. Our dataset contains 220 images which cover six

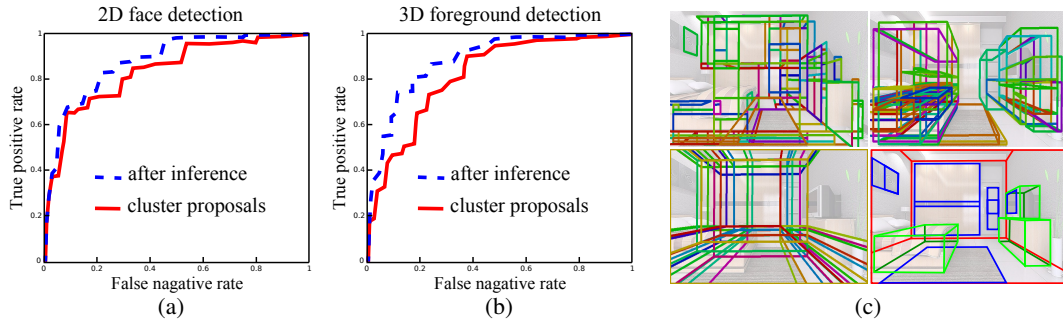

Figure 6: Quantitative performance of 2D face detection (a) and 3D foreground detection (b) in our dataset. (c) An example of the top proposals and the result after inference.

indoor scene categories: bedroom, living room, kitchen, classroom, office room, and corridor. The dataset is available on the project webpage[1]. The ground-truths are hand labeled segments for scene components for each image. Our algorithm usually takes 20s in clustering, 40s in sampling, and 1m in preparing input features.

**Qualitative evaluation**: The experimental results in Fig.7 is obtained by applying different production rules to images in our dataset. With the AND rules only, the algorithm obtains reasonable results and successfully recovers some salient 3D foreground objects and 2D faces. With both the AND and SET rules, the cooperative "+" relations help detect some weak visual entities. Fig.8 lists more experimental results of the UIUC dataset. The proposed algorithm recovers most of the indoor components. In the last row, we show some challenging images with missing detections and false positives. Weak line information, ambiguous overlapping objects, salient patterns and clustered structures would confuse our algorithm.

**Quantitative evaluation**: We first evaluate the detection of 2D faces, 3D foreground objects in our dataset. The detection error is measured on the pixel level, it indicates how many pixels are correctly labelled. In Fig.6, the red curves show the ROC of 2D faces / 3D objects detection in clustering stage. They are computed by thresholding cluster probabilities given by Eq.3. The blue curves show the ROC of final detection given a partial parse tree after MCMC inference. They are computed by thresholding the marginal probability given Eq.2. Using the UIUC dataset, we compare our algorithm to four other state-of-the-art indoor scene parsing algorithms, Hoiem et al. [1], Hedau et al. [2], Wang et al. [3] and Lee et al. [4]. All of these four algorithms used discriminative learning of Structure-SVM (or Latent-SVM). By applying the production rules and the contextual relations, our generative grammar model outperforms others as shown in Table.1.

## 6  Conclusion

In this paper, we propose a framework of geometric image parsing using Stochastic Scene Grammar (SSG). The grammar model is used to represent the compositional structure of visual entities. It is beyond the traditional probabilistic context-free grammars (PCFGs) in a few aspects: spatial context, production rules for multiple occurrences of objects, richer image appearance and geometric properties. We also design a hierarchical cluster sampling algorithm that uses contextual relations to accelerate the Markov chain search. The SSG model is flexible to model other compositional structures by applying different production rules and contextual relations. An interesting extension of our work can be adding semantic labels, such as chair, desk, shelf etc., to 3D objects. This will be interesting to discover new relations between TV and sofa, desk and chair, bed and night table as demonstrated in [26].

### Acknowledgments

The work is supported by grants from NSF IIS-1018751, NSF CNS-1028381 and ONR MURI N00014-10-1-0933.

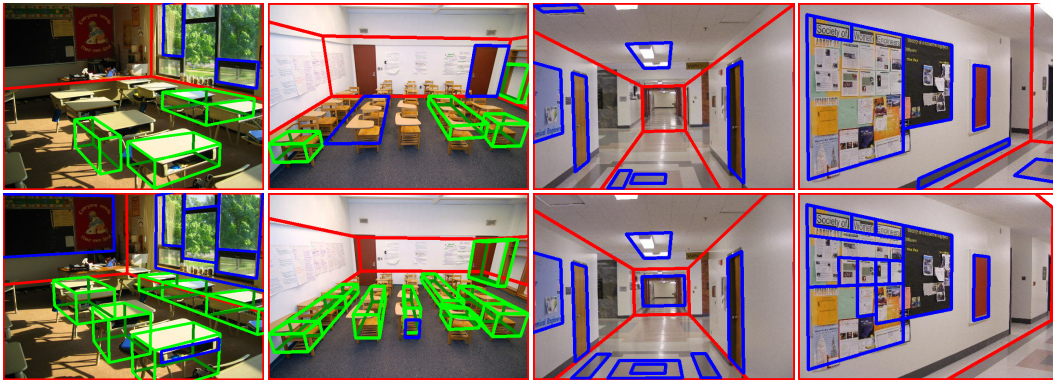

Figure 7: Experimental results by applying the AND/OR rules (the first row) and applying all AND/OR/SET rules (the second row) in our dataset

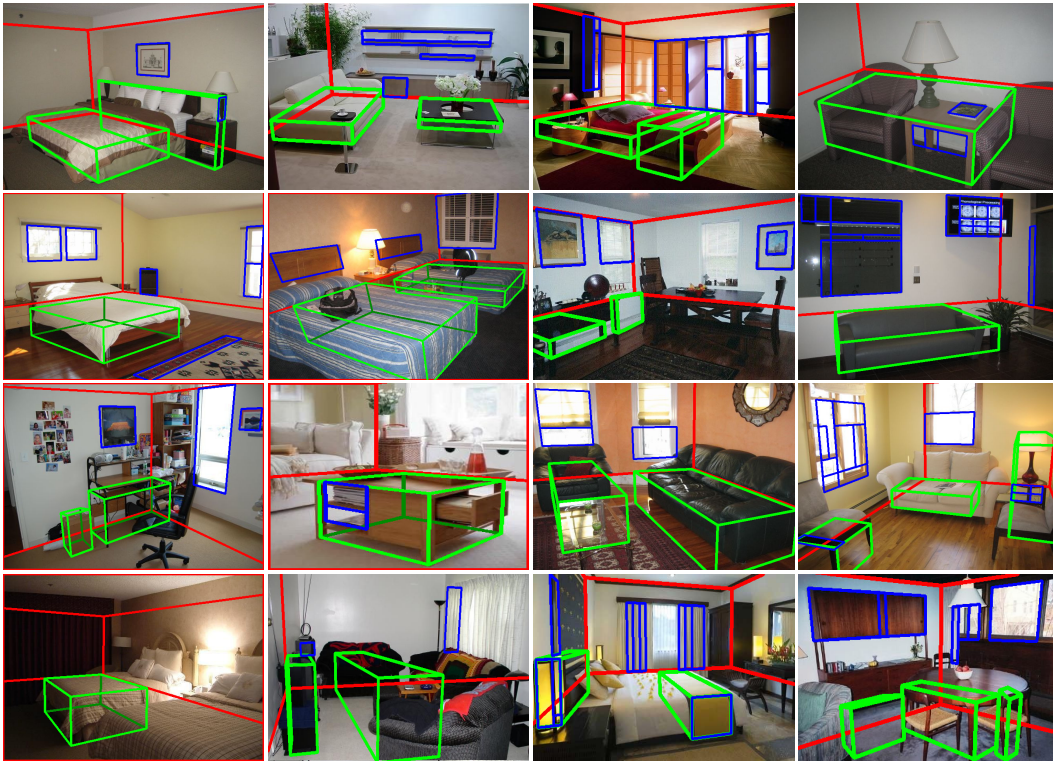

Figure 8: Experimental results of more complex indoor images in UIUC dataset [2]. The last row shows some challenging images with missing detections and false positives of proposed algorithm.

Table 1: Segmentation precision compared with Hoiem et al. 2007 [1], Hedau et al. 2009 [2], Wang et al. 2010 [3] and Lee et al. 2010 [4] in the UIUC dataset [2].

| Segmentation precision | [1] | [2] | [3] | [4] | Our method |
|---|---|---|---|---|---|
| Without rules | 73.5% | 78.8% | 79.9% | 81.4% | 80.5% |
| With 3D "-" constraints | - | - | - | 83.8% | 84.4% |
| With AND, OR rules | - | - | - | - | 85.1% |
| With AND, OR, SET rules | - | - | - | - | 85.5% |

## Footnotes

[1] http://www.stat.ucla.edu/~ybzhao/research/sceneparsing

## References

[1] Hoiem, D., Efors, A., & Hebert, M. (2007) Recovering Surface Layout from an Image *IJCV* 75(1).

[2] Hedau, V., Hoiem, D., & Forsyth, D. (2009) Recovering the spatial layout of cluttered rooms. *In ICCV.*

[3] Wang, H., Gould, S. & Koller, D. (2010) Discriminative Learning with Latent Variables for Cluttered Indoor Scene Understanding. *ECCV.*

[4] Lee, D., Gupta, A. Hebert, M., & Kanade, T. (2010) Estimating Spatial Layout of Rooms using Volumetric Reasoning about Objects and Surfaces *Advances in Neural Information Processing Systems 7*, pp. 609-616. Cambridge, MA: MIT Press.

[5] Shotton, J., & Winn, J. (2007) TextonBoost for Image Understanding: Multi-Class Object Recognition and Segmentation by Jointly Modeling Texture, Layout, and Context. *IJCV*

[6] Tu, Z., & Bai, X. (2009) Auto-context and Its Application to High-level Vision Tasks and 3D Brain Image Segmentation *PAMI*

[7] Lafferty, J. D., McCallum, A., & Pereira, F. C. N. (2001). Conditional random fields: probabilistic models for segmenting and labeling sequence data. *In ICML* (pp. 282-289).

[8] Saxena, A., Sun, M. & Ng, A. (2008) Make3d: Learning 3D scene structure from a single image. *PAMI.*

[9] Gupta, A., Efros,A., & Hebert, M. (2010) Blocks World Revisited: Image Understanding using Qualitative Geometry and Mechanics. *ECCV.*

[10] Tsochantaridis, T. Joachims, T. Hofmann & Y. Altun (2005) Large Margin Methods for Structured and Interdependent Output Variables, *JMLR*, Vol. 6, pages 1453-1484.

[11] Manning, C., & Schuetze, H. (1999) Foundations of statistical natural language processing. *Cambridge: MIT Press.*

[12] Chen, H., Xu, Z., Liu, Z., & Zhu, S. C. (2006) Composite templates for cloth modeling and sketching. *In CVPR* (1) pp. 943-950.

[13] Jin, Y., & Geman, S. (2006) Context and hierarchy in a probabilistic image model. *In CVPR* (2) pp. 2145-2152.

[14] Zhu, L., & Yuille, A. L. (2005) A hierarchical compositional system for rapid object detection. *Advances in Neural Information Processing Systems 7*, pp. 609-616. Cambridge, MA: MIT Press.

[15] Fidler, S., & Leonardis, A. (2007) Towards Scalable Representations of Object Categories: Learning a Hierarchy of Parts. *In CVPR.*

[16] Zhu, S. C., & Mumford, D. (2006) A stochastic grammar of images. *Foundations and Trends in Computer Graphics and Vision*, 2(4), 259-362.

[17] Johnson, M., Griffiths, T. L, & Goldwater, S. (2007) Adaptor Grammars: A Framework for Specifying Compositional Nonparametric Bayesian Models. In G. Tesauro, D. S. Touretzky and T.K. Leen (eds.), *Advances in Neural Information Processing Systems 7*, pp. 609-616. Cambridge, MA: MIT Press.

[18] Han, F., & Zhu, S. C. (2009) Bottom-Up/Top-Down Image Parsing with Attribute Grammar *PAMI*

[19] Porway, J., & Zhu, S. C. (2010) Hierarchical and Contextual Model for Aerial Image Understanding. *Int'l Journal of Computer Vision*, vol.88, no.2, pp 254-283.

[20] Porway, J., & Zhu, S. C. (2011) C4 : Computing Multiple Solutions in Graphical Models by Cluster Sampling. *PAMI*, vol.33, no.9, 1713-1727.

[21] Lee, D., Hebert, M., & Kanade, T. (2009) Geometric Reasoning for Single Image Structure Recovery *In CVPR.*

[22] Hedau, V., Hoiem, D., & Forsyth, D. (2010). Thinking Inside the Box: Using Appearance Models and Context Based on Room Geometry. *In ECCV.*

[23] Felzenszwalb, P.F. (2010) Cascade Object Detection with Deformable Part Models. *In CVPR.*

[24] Pero, L. D., Guan, J., Brau, E. Schlecht, J. & Barnard, K. (2011) Sampling Bedrooms. *In CVPR.*

[25] Zhu, S. C., Wu, Y., & Mumford, D. (1997) Minimax Entropy Principle and Its Application to Texture Modeling. *Neural Computation* 9(8): 1627-1660.

[26] Yu, L. F., Yeung, S. K., Tang, C. K., Terzopoulos, D., Chan, T. F. & Osher, S. (2011) Make it home: automatic optimization of furniture arrangement. *ACM Transactions on Graphics* 30(4): pp.86

